# Competitive Anti-Hebbian Learning of Invariants

**Nicol N. Schraudolph**
Computer Science & Engr. Dept.
University of California, San Diego
La Jolla, CA 92093–0114
*nici@cs.ucsd.edu*

**Terrence J. Sejnowski**
Computational Neurobiology Laboratory
The Salk Institute for Biological Studies
La Jolla, CA 92186-5800
*tsejnowski@ucsd.edu*

## Abstract

Although the detection of *invariant* structure in a given set of input patterns is vital to many recognition tasks, connectionist learning rules tend to focus on directions of high variance *(principal components)*. The *prediction paradigm* is often used to reconcile this dichotomy; here we suggest a more direct approach to invariant learning based on an *anti-Hebbian* learning rule. An unsupervised two-layer network implementing this method in a competitive setting learns to extract coherent depth information from random-dot stereograms.

## 1 INTRODUCTION: LEARNING INVARIANT STRUCTURE

Many connectionist learning algorithms share with principal component analysis (Jolliffe, 1986) the strategy of extracting the directions of highest variance from the input. A single Hebbian neuron, for instance, will come to encode the input's first principal component (Oja and Karhunen, 1985); various forms of lateral interaction can be used to force a layer of such nodes to differentiate and span the principal component subspace — cf. (Sanger, 1989; Kung, 1990; Leen, 1991), and others. The same type of representation also develops in the hidden layer of backpropagation autoassociator networks (Baldi and Hornik, 1989).

However, the directions of highest variance need not always be those that yield the most information, or — as the case may be — the information we are interested in (Intrator, 1991). In fact, it is sometimes desirable to extract the *invariant structure* of a stimulus instead, learning to encode those aspects that vary the least. The problem, then, is how to achieve this within a connectionist framework that is so closely tied to the maximization of variance.

In (Földiák, 1991), spatial invariance is turned into a temporal feature by presenting *transformation sequences* within invariance classes as a stimulus. A built-in temporal smoothness constraint enables Hebbian neurons to learn these transformations, and hence the invariance classes. Although this is an efficient and neurobiologically attractive strategy it is limited by its strong assumptions about the nature of the stimulus.

A more general approach is to make information about invariant structure available in the error signal of a supervised network. The most popular way of doing this is to require the network to predict the next patch of some structured input from the preceding context, as in (Elman, 1990); the same prediction technique can be used across space as well as time. It is also possible to explicitly derive an error signal from the mutual information between two patches of structured input (Becker and Hinton, 1992), a technique which has been applied to viewpoint-invariant object recognition (Zemel and Hinton, 1991).

## 2   METHODS

### 2.1   ANTI-HEBBIAN FEEDFORWARD LEARNING

In most formulations of the covariance learning rule it is quietly assumed that the learning rate be positive. By reversing the sign of this constant in a recurrent autoassociator, Kohonen constructed a "novelty filter" that learned to be insensitive to familiar features in its input (Kohonen, 1989). More recently, such anti-Hebbian synapses have been used for lateral decorrelation of feature detectors (Barlow and Földiák, 1989; Leen, 1991) as well as — in differential form — removal of temporal variations from the input (Mitchison, 1991).

We suggest that in certain cases the use of anti-Hebbian feedforward connections to learn invariant structure may eliminate the need to bring in the heavy machinery of supervised learning algorithms required by the prediction paradigm, with its associated lack of neurobiological plausibility. Specifically, this holds for linear problems, where the stimuli lie near a hyperplane in the input space: the weight vector of an anti-Hebbian neuron will move into a direction normal to that hyperplane, thus characterizing the invariant structure.

Of course a set of Hebbian feature detectors whose weight vectors span the hyperplane would characterize the associated class of stimuli just as well. The anti-Hebbian learning algorithm, however, provides a more efficient representation when the dimensionality of the hyperplane is more than half that of the input space, since less normal vectors than spanning vectors are required for unique characterization in this case. Since they remove rather than extract the variance within a stimulus class, anti-Hebbian neurons also present a very different output representation to subsequent layers.

Unfortunately it is not sufficient to simply negate the learning rate of a layer of Hebbian feature detectors in order to turn them into working anti-Hebbian invariance detectors: although such a change of sign does superficially achieve the intended effect, many of the subtleties that make Hebb's rule work in practice do not survive the transformation. In what follows we address some of the problems thus introduced.

Like the Hebb rule, anti-Hebbian learning requires weight normalization, in this case to prevent weight vectors from collapsing to zero. Oja's active decay rule (Oja, 1982) is a popular local approximation to explicit weight normalization:

$$\Delta \vec{w} = \eta(\vec{x}y - \vec{w}y^2), \text{ where } y = \vec{w}^T \vec{x} \qquad (1)$$

Here the first term in parentheses represents the standard Hebb rule, while the second is the active decay. Unfortunately, Oja's rule can not be used for weight growth in anti-Hebbian neurons since it is unstable for negative learning rates ($\eta < 0$), as is evident from the observation that the growth/decay term is proportional to $\vec{w}$. In our experiments, explicit $L_2$-normalization of weight vectors was therefore used instead.

Hebbian feature detectors attain maximal activation for the class of stimuli they represent. Since the weight vectors of anti-Hebbian invariance detectors are *normal* to the invariance class they represent, membership in that class is signalled by a zero activation. In other words, linear anti-Hebbian nodes signal *violations* of the constraints they encode rather than compliance. While such an output representation can be highly desirable for some applications[1], it is unsuitable for others, such as the classification of mixtures of invariants described below.

We therefore use a symmetric activation function that responds maximally for a zero net input, and decays towards zero for large net inputs. More specifically, we use Gaussian activation functions, since these allow us to interpret the nodes' outputs as class membership probabilities. Soft competition between nodes in a layer can then be implemented simply by normalizing these probabilities (i.e. dividing each output by the sum of outputs in a layer), then using them to scale weight changes (Nowlan, 1990).

## 2.2  AN ANTI-HEBBIAN OBJECTIVE FUNCTION

The magnitude of weight change in a Hebbian neuron is proportional to the cosine of the angle between input and weight vectors. This means that nodes that best represent the current input learn faster than those which are further away, thus encouraging differentiation among weight vectors. Since anti-Hebbian weight vectors are normal to the hyperplanes they represent, those that best encode a given stimulus will experience the *least* change in weights. As a result, weight vectors will tend to clump together unless weight changes are rescaled to counteract this deficiency. In our experiments, this is done by the soft competition mechanism; here we present a more general framework towards this end.

A simple Hebbian neuron maximizes the variance of its output $y$ through stochastic approximation by performing gradient ascent in $\frac{1}{2}y^2$ (Oja and Karhunen, 1985):

$$\Delta w_i \propto \frac{\partial}{\partial w_i} \frac{1}{2} y^2 = y \frac{\partial}{\partial w_i} y = x_i y \qquad (2)$$

As seen above, it is not sufficient for an anti-Hebbian neuron to simply perform gradient descent in the same function. Instead, an objective function whose derivative has inverse magnitude to the above at every point is needed, as given by

$$\Delta w_i \propto \frac{\partial}{\partial w_i} \frac{1}{2} \log(y^2) = \frac{1}{y} \frac{\partial}{\partial w_i} y = \frac{x_i}{y} \qquad (3)$$

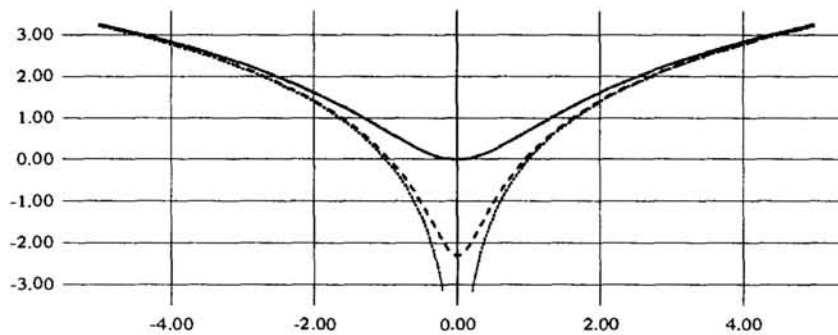

Figure 1: Possible objective functions for anti-Hebbian learning (see text).

Unfortunately, the pole at $y = 0$ presents a severe problem for simple gradient descent methods: the near-infinite derivatives in its vicinity lead to catastrophically large step sizes. More sophisticated optimization methods deal with this problem by explicitly controlling the step size; for plain gradient descent we suggest reshaping the objective function at the pole such that its partials never exceed the input in magnitude:

$$\Delta w_i \propto \frac{\partial}{\partial w_i} \varepsilon \log(y^2 + \varepsilon^2) = \frac{2\varepsilon x_i y}{y^2 + \varepsilon^2}, \tag{4}$$

where $\varepsilon > 0$ is a free parameter determining at which point the logarithmic slope is abandoned in favor of a quadratic function which forms an optimal trapping region for simple gradient descent (Figure 1).

## 3    RESULTS ON RANDOM-DOT STEREOGRAMS

In random-dot stereograms, stimuli of a given stereo disparity lie on a hyperplane whose dimensionality is half that of the input space plus the disparity in pixels. This is easily appreciated by considering that given, say, the left half-image and the disparity, one can predict the right half-image except for the pixels shifted in at the edge. Thus stereo disparities that are small compared to the receptive field width can be learned equally well by Hebbian and anti-Hebbian algorithms; when the disparity approaches receptive field width, however, anti-Hebbian neurons have a distinct advantage.

### 3.1    SINGLE LAYER NETWORK: LOCAL DISPARITY TUNING

Our training set consisted of stereo images of 5,000 frontoparallel strips at uniformly random depth covered densely with Gaussian features of random location, width, polarity and power. The images were discretized by integrating over pixel bins in order to allow for sub-pixel disparity acuity. Figure 2 shows that a single cluster of five anti-Hebbian nodes with soft competition develops near-perfect tuning curves for local stereo disparity after 10 sweeps through this training set. This disparity tuning is achieved by learning to have corresponding weights (at the given disparity) be of equal magnitude but opposite sign, so that any stimulus pattern at that disparity yields a zero net input and thus maximal response.

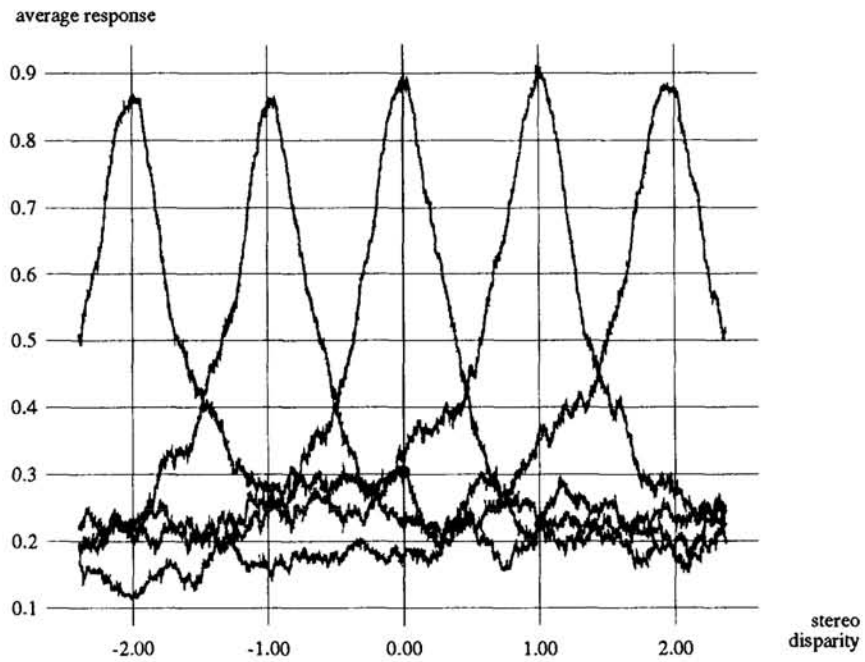

Figure 2: Sliding window average response of first-layer nodes after presentation of 50,000 stereograms as a function of stimulus disparity: strong disparity tuning is evident.

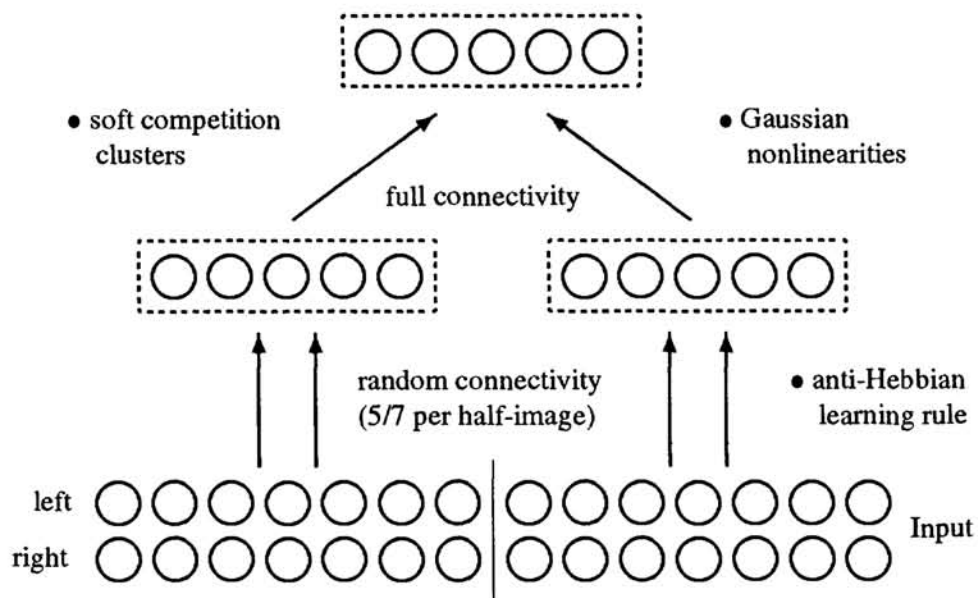

Figure 3: Architecture of the network (see text).

Note, however, that this type of detector suffers from false positives: input patterns that happen to yield near-zero net input even though they have a different stereo disparity. Although the individual response of a tuned node to an input pattern of the wrong disparity is therefore highly idiosyncratic, the sliding window average of each response with its 250 closest neighbors (with respect to disparity) shown in Figure 2 is far more well-behaved. This indicates that the average activity over a number of patterns (in a "moving stereogram" paradigm) — or, alternatively, over a population of nodes tuned to the same disparity — allows discrimination of disparities with sub-pixel accuracy.

## 3.2    TWO-LAYER NETWORK: COHERENT DISPARITY TUNING

In order to investigate the potential for hierarchical application of this architecture, it was extended to two layers as shown in Figure 3. The two first-layer clusters with non-overlapping receptive fields extract local stereo disparity as before; their output is monitored by a second-layer cluster. Note that there is no backpropagation of derivatives: all three clusters use the same unsupervised learning algorithm.

This network was trained on coherent input, i.e. stimuli for which the stereo disparity was identical across the receptive field boundary of first-layer clusters. As shown in Figure 4, the second layer learns to preserve the first layer's disparity tuning for coherent patterns, albeit in in somewhat degraded form. Each node in the second layer learns to pick out exactly the two corresponding nodes in the first-layer clusters, again by giving them weights of equal magnitude but opposite sign.

However, the second layer represents more than just a noisy copy of the first layer: it meaningfully integrates coherence information from the two receptive fields. This can be demonstrated by testing the trained network on non-coherent stimuli which exhibit a depth discontinuity between the receptive fields of first-layer clusters. The overall response of the second layer is tuned to the coherent stimuli it was trained on (Figure 5).

## 4    DISCUSSION

Although a negation of the learning rate introduces various problems to the Hebb rule, feedforward anti-Hebbian networks can pick up invariant structure from the input. We have demonstrated this in a competitive classification setting; other applications of this framework are possible. We find the *subsumption* aspect of anti-Hebbian learning particularly intriguing: the real world is so rich in redundant data that a learning rule which can adaptively *ignore* much of it must surely be an advantage. From this point of view, the promising first experiments we have reported here use quite impoverished inputs; one of our goals is therefore to extend this work towards real-world stimuli.

**Acknowledgements**

We would like to thank Geoffrey Hinton, Sue Becker, Tony Bell and Steve Nowlan for the stimulating and helpful discussions we had. Special thanks to Sue Becker for permission to use her random-dot stereogram generator early in our investigation. This work was supported by a fellowship stipend from the McDonnell-Pew Center for Cognitive Neuro-science at San Diego to the first author, who also received a NIPS travel grant enabling him to attend the conference.

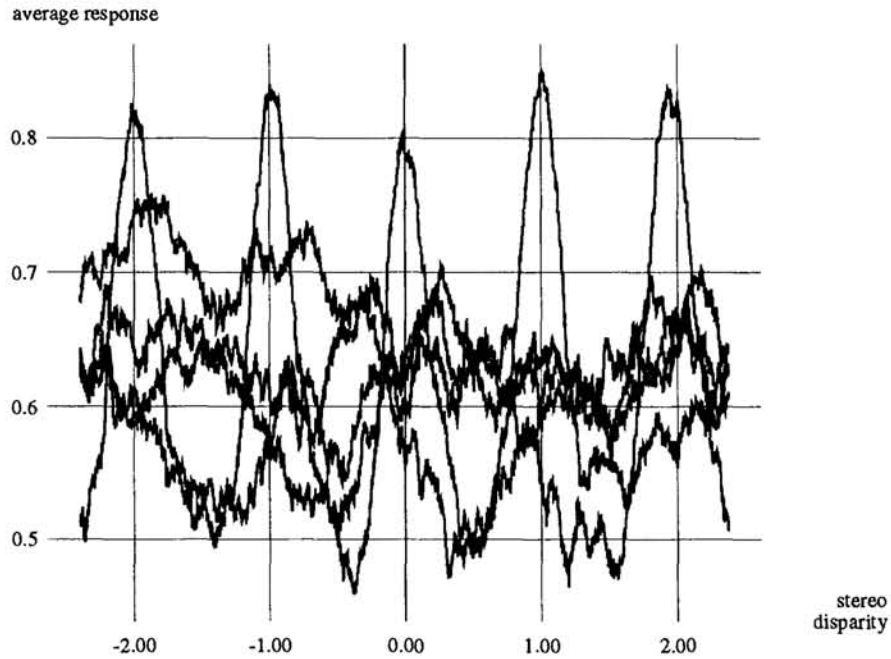

Figure 4: Sliding window average response of second-layer nodes after presentation of 250,000 coherent stereograms as a function of stimulus disparity: disparity tuning is preserved in degraded form.

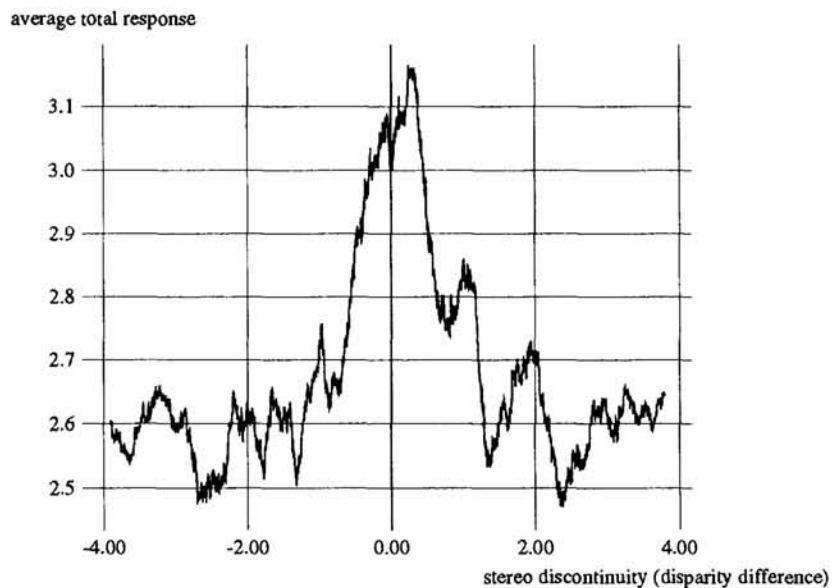

Figure 5: Sliding window average of total second-layer response to non-coherent input as a function of stimulus discontinuity: second layer is tuned to coherent patterns.

**References**

Baldi, P. and Hornik, K. (1989). Neural networks and principal component analysis: Learning from examples without local minima. *Neural Networks*, 2:53–58.

Barlow, H. B. and Földiák, P. (1989). Adaptation and decorrelation in the cortex. In Durbin, R. M., Miall, C., and Mitchison, G. J., editors, *The Computing Neuron*, chapter 4, pages 54–72. Addison-Wesley, Wokingham.

Becker, S. and Hinton, G. E. (1992). A self-organizing neural network that discovers surfaces in random-dot stereograms. *Nature*, to appear.

Elman, J. (1990). Finding structure in time. *Cognitive Science*, 14:179–211.

Földiák, P. (1991). Learning invariance from transformation sequences. *Neural Computation*, 3:194–200.

Intrator, N. (1991). Exploratory feature extraction in speech signals. In (Lippmann et al., 1991), pages 241–247.

Jolliffe, I. (1986). *Principal Component Analysis*. Springer-Verlag, New York.

Kohonen, T. (1989). *Self-Organization and Associative Memory*. Springer-Verlag, Berlin, 3 edition.

Kung, S. Y. (1990). Neural networks for extracting constrained principal components. submitted to *IEEE Trans. Neural Networks*.

Leen, T. K. (1991). Dynamics of learning in linear feature-discovery networks. *Network*, 2:85–105.

Lippmann, R. P., Moody, J. E., and Touretzky, D. S., editors (1991). *Advances in Neural Information Processing Systems*, volume 3, Denver 1990. Morgan Kaufmann, San Mateo.

Mitchison, G. (1991). Removing time variation with the anti-hebbian differential synapse. *Neural Computation*, 3:312–320.

Nowlan, S. J. (1990). Maximum likelihood competitive learning. In Touretzky, D. S., editor, *Advances in Neural Information Processing Systems*, volume 2, pages 574–582, Denver 1989. Morgan Kaufmann, San Mateo.

Oja, E. (1982). A simplified neuron model as a principal component analyzer. *Journal of Mathematical Biology*, 15:267–273.

Oja, E. and Karhunen, J. (1985). On stochastic approximation of the eigenvectors and eigenvalues of the expectation of a random matrix. *Journal of Mathematical Analysis and Applications*, 106:69–84.

Sanger, T. D. (1989). Optimal unsupervised learning in a single-layer linear feedforward neural network. *Neural Networks*, 2:459–473.

Zemel, R. S. and Hinton, G. E. (1991). Discovering viewpoint-invariant relationships that characterize objects. In (Lippmann et al., 1991), pages 299–305.

## Footnotes

[1]Consider the *subsumption architecture* of a hierarchical network in which higher layers only receive information that is not accounted for by earlier layers.
